# Causal discovery in multiple models from different experiments

**Tom Claassen**
Radboud University Nijmegen
The Netherlands
tomc@cs.ru.nl

**Tom Heskes**
Radboud University Nijmegen
The Netherlands
tomh@cs.ru.nl

## Abstract

A long-standing open research problem is how to use information from different experiments, including background knowledge, to infer causal relations. Recent developments have shown ways to use multiple data sets, provided they originate from *identical* experiments. We present the MCI-algorithm as the first method that can infer provably valid causal relations in the large sample limit from *different* experiments. It is fast, reliable and produces very clear and easily interpretable output. It is based on a result that shows that constraint-based causal discovery is decomposable into a candidate pair identification and subsequent elimination step that can be applied separately from different models. We test the algorithm on a variety of synthetic input model sets to assess its behavior and the quality of the output. The method shows promising signs that it can be adapted to suit causal discovery in real-world application areas as well, including large databases.

## 1 Introduction

Discovering causal relations from observational data is an important, ubiquitous problem in science. In many application areas there is a multitude of data from different but related experiments. Often the set of measured variables is not the same between trials, or the circumstances under which they were conducted differed, making it difficult to compare and evaluate results, especially when they seem to contradict each other, e.g. when a certain dependency is observed in one experiment, but not in another. Results obtained from one data set are often used to either corroborate or challenge results from another. Yet how to reconcile information from multiple sources, including background knowledge, into a single, more informative model is still an open problem.

Constraint-based methods like the FCI-algorithm [1] are provably correct in the large sample limit, as are Bayesian methods like the greedy search algorithm GES [2] (with additional post-processing steps to handle hidden confounders). Both are defined in terms of modeling a single data set and have no principled means to relate to results from other sources in the process. Recent developments, like the ION-algorithm by Tillman et al. [3], have shown that it is possible to integrate multiple, partially overlapping data sets. However, such algorithms are still essentially single model learners in the sense that they assume there is one, single encapsulating structure that accounts for *all* observed dependencies in the different models. In practice, observed dependencies often differ between data sets, precisely because the experimental circumstances were not identical in different experiments, even when the causal system at the heart of it was the same. The method we develop in this article shows how to distinguish between causal dependencies internal to the system under investigation and merely contextual dependencies.

Mani et al. [4] recognized the 'local' aspect of causal discovery from Y-structures embedded in data: it suffices to establish a certain (in)dependency pattern between variables, without having to uncover the entire graph. In section 4 we take this one step further by showing that such causal

discovery can be decomposed into two separate steps: a conditional independency to identify a pair of possible causal relations (one of which is true), and then a conditional dependency that eliminates one option, leaving the other. The two steps rely only on local (marginal) aspects of the distribution. As a result the conclusion remains valid, even when, unlike causal inference from Y-structures, the two pieces of information are taken from different models. This forms the basis underpinning the MCI-algorithm in section 6.

Section 2 of this article introduces some basic terminology. Section 3 models different experiments. Section 4 establishes the link between conditional independence and local causal relations, which is used in section 5 to combine multiple models into a single causal graph. Section 6 describes a practical implementation in the form of the MCI-algorithm. Sections 7 and 8 discuss experimental results and suggest possible extensions to other application areas.

## 2 Graphical model preliminaries

First a few familiar notions from graphical model theory used throughout the article. A *directed graph* $\mathcal{G}$ is a pair $\langle \mathbf{V}, \mathbf{E} \rangle$, where $\mathbf{V}$ is a set of vertices or nodes and $\mathbf{E}$ is a set of edges between pairs of nodes, represented by arrows $X \to Y$. A *path* $\pi = \langle V_0, \ldots, V_n \rangle$ between $V_0$ and $V_n$ in $\mathcal{G}$ is a sequence of distinct vertices such that for $0 \leq i \leq n-1$, $V_i$ and $V_{i+1}$ are connected by an edge in $\mathcal{G}$. A *directed path* is a path that is traversed entirely in the direction of the arrows. A *directed acyclic graph* (DAG) is a directed graph that does not contain a directed path from any node to itself. A vertex $X$ is an *ancestor* of $Y$ (and $Y$ is a *descendant* of $X$) if there is a directed path from $X$ to $Y$ in $\mathcal{G}$ or if $X = Y$. A vertex $Z$ is a *collider* on a path $\pi = \langle \ldots, X, Z, Y, \ldots \rangle$ if it contains the subpath $X \to Z \leftarrow Y$, otherwise it is a *noncollider*. A *trek* is a path that does not contain any collider.

For disjoint (sets of) vertices $X$, $Y$ and $\mathbf{Z}$ in a DAG $\mathcal{G}$, $X$ is *d-connected* to $Y$ conditional on $\mathbf{Z}$ (possibly empty), iff there exists an unblocked path $\pi = \langle X, \ldots, Y \rangle$ between $X$ and $Y$ given $\mathbf{Z}$, i.e. such that every collider on $\pi$ is an ancestor of some $Z \in \mathbf{Z}$ and every noncollider on $\pi$ is not in $\mathbf{Z}$. If not, then all such paths are blocked, and $X$ is said to be *d-separated* from $Y$; see [5, 1] for details.

**Definition 1.** Two nodes $X$ and $Y$ are *minimally conditionally independent* given a set of nodes $\mathbf{Z}$, denoted $[X \perp\!\!\!\perp Y \mid \mathbf{Z}]$, iff $X$ is conditionally independent of $Y$ given a *minimal* set of nodes $\mathbf{Z}$. Here minimal, indicated by the square brackets, implies that the relation does not hold for any proper subset $\mathbf{Z}' \subsetneq \mathbf{Z}$ of the (possibly empty) set $\mathbf{Z}$.

A *causal DAG* $\mathcal{G}_C$ is a graphical model in the form of a DAG where the arrows represent direct causal interactions between variables in a system [6]. There is a causal relation $X \Rightarrow Y$, iff there is a directed path from $X$ to $Y$ in $\mathcal{G}_C$. Absence of such a path is denoted $X \not\Rightarrow Y$. The *causal Markov condition* links the structure of a causal graph to its probabilistic concomitant, [5]: two variables $X$ and $Y$ in a causal DAG $\mathcal{G}_C$ are dependent given a set of nodes $\mathbf{Z}$, iff they are connected by a path $\pi$ in $\mathcal{G}_C$ that is unblocked given $\mathbf{Z}$; so there is a dependence $X \not\perp\!\!\!\perp Y$ iff there is a trek between $X$ and $Y$ in the causal DAG.

We assume that the systems we consider correspond to some underlying causal DAG over a great many observed and unobserved nodes. The distribution over the subset of observed variables can then be represented by a *(maximal) ancestral graph* (MAG) [7]. Different MAGs can represent the same distribution, but only the invariant features, common to all MAGs that can faithfully represent that distribution, carry identifiable causal information. The *complete partial ancestral graph* (CPAG) $\mathcal{P}$ that represents the equivalence class $[\mathcal{G}]$ of a MAG $\mathcal{G}$ is a graph with either a tail '$-$', arrowhead '>' or circle mark '∘' at each end of an edge, such that $\mathcal{P}$ has the same adjacencies as $\mathcal{G}$, and there is a tail or arrowhead on an edge in $\mathcal{P}$ iff it is invariant in $[\mathcal{G}]$, otherwise it has a circle mark [8]. The CPAG of a given MAG is unique and maximally informative for $[\mathcal{G}]$. We use CPAGs as a concise and intuitive graphical representation of all conditional (in)dependence relations between nodes in an observed distribution; see [7, 8] for more information on how to read independencies directly from a MAG/CPAG using the *m*-separation criterion, which is essentially just the *d*-separation criterion, only applied to MAGs.

Throughout this article we also adopt the *causal faithfulness assumption*, which implies that all and only the conditional independence relations entailed by the causal Markov condition applied to the true causal DAG will hold in the joint probability distribution over the variables in $\mathcal{G}_C$. For an in-depth discussion of the justification and connection between these assumptions, see [9].

## 3 Modeling the system

Random variation in a system corresponds to the impact of unknown external variables, see [5]. Some of these external factors may be actively controlled, e.g. in clinical trials, or passively observed as the natural embedding of a system in its environment. We refer to both observational and controlled studies as *experiments*. External factors that affect two or more variables in a system simultaneously, can lead to dependencies that are not part of the system. Different external factors may bring about observed dependencies that differ between models, seemingly contradicting each other. By modeling this external environment explicitly as a set of unobserved (hypothetical) context nodes that causally affect the system under scrutiny we can account for this effect.

**Definition 2.** The *external context* $\mathcal{G}_E$ of a causal DAG $\mathcal{G}_C$ is a set of independent nodes $\mathbf{U}$ in combination with links from every $U \in \mathbf{U}$ to one or more nodes in $\mathcal{G}_C$. The total causal structure of an experiment then becomes $\mathcal{G}_T = \{\mathcal{G}_E + \mathcal{G}_C\}$.

Figure 1 depicts a causal system in three different experiments (double lined arrows indicate direct causal relations; dashed circles represent unobserved variables). The second and third experiment will result in an observed dependency between variables $A$ and $B$, whereas the first one will not. The context only introduces arrows from nodes in $\mathcal{G}_E$ to $\mathcal{G}_C$ which can never result in a cycle, therefore the structure of an experiment $\mathcal{G}_T$ is also a causal DAG. Note that differences in dependencies can only arise from different *structures* of the external context.

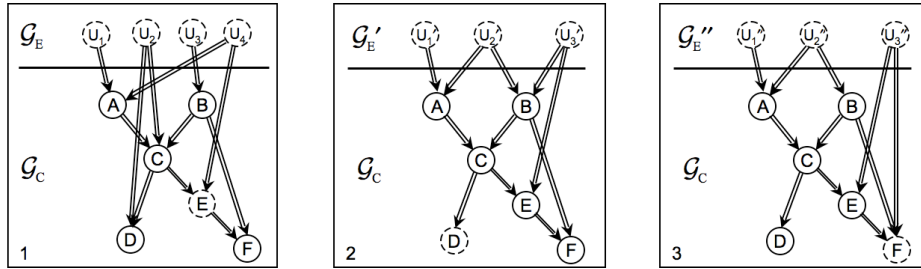

Figure 1: A causal system $\mathcal{G}_C$ in different experiments

In this paradigm different experiments become variations in context of a constant causal system. The goal of causal discovery from multiple models can then be stated as: "Given experiments with unknown total causal structures $\mathcal{G}_T = \{\mathcal{G}_E + \mathcal{G}_C\}$, $\mathcal{G}'_T = \{\mathcal{G}'_E + \mathcal{G}_C\}$, etc., and known joint probability distributions $P(\mathbf{V} \subset \mathcal{G}_T)$, $P'(\mathbf{V}' \subset \mathcal{G}'_T)$, etc., which variables are connected by a directed path in $\mathcal{G}_C$?". We assume that the large sample limit distributions $P(\mathbf{V})$ are known and can be used to obtain categorical statements about probabilistic (in)dependencies between sets of nodes. Finally, we will assume there is no selection bias, see [10], nor blocking interventions on $\mathcal{G}_C$, as accounting for the impact would unnecessarily complicate the exposition.

## 4 Causal relations in arbitrary context

A remarkable result that, to the best of our knowledge, has not been noted before, is that a minimal conditional independence *always* implies the presence of a causal relation. (*See appendix for an outline of all proofs in this article.*)

**Theorem 1.** Let $X$, $Y$, $\mathbf{Z}$ and $W$ be four disjoint (sets of) nodes (possibly empty) in an experiment with causal structure $\mathcal{G}_T = \{\mathcal{G}_E + \mathcal{G}_C\}$, then the following rules apply, for arbitrary $\mathcal{G}_E$

(1) a *minimal* conditional independence $[X \perp\!\!\!\perp Y \mid \mathbf{Z}]$ implies causal links $Z \Rightarrow X$ and/or $Z \Rightarrow Y$ from every $Z \in \mathbf{Z}$ to $X$ and/or $Y$ in $\mathcal{G}_C$,

(2) a conditional dependence $X \not\perp\!\!\!\perp Y \mid \mathbf{Z} \cup W$ induced by a node $W$, i.e. with $X \perp\!\!\!\perp Y \mid \mathbf{Z}$, implies that there are no causal links $W \not\Rightarrow X$, $W \not\Rightarrow Y$ or $W \not\Rightarrow Z$ for any $Z \in \mathbf{Z}$ in $\mathcal{G}_C$,

(3) a conditional independence $X \perp\!\!\!\perp Y \mid \mathbf{Z}$ implies the absence of (direct) causal paths $X \Rightarrow Y$ or $X \Leftarrow Y$ in $\mathcal{G}_C$ between $X$ and $Y$ that are not mediated by nodes in $\mathbf{Z}$.

The theorem establishes independence patterns that signify (absence of) a causal origin, independent of the (unobserved) external background. Rule (1) identifies a candidate pair of causal relations from a conditional independence. Rule (2) identifies the absence of causal paths from unshielded colliders in $\mathcal{G}$, see also [1]. Rule (3) eliminates direct causal links between variables.

The final step towards causal discovery from multiple models now takes a surprisingly simple form:

**Lemma 1.** Let $X, Y$ and $Z \in \mathbf{Z}$ be disjoint (sets of) variables in an experiment with causal structure $\mathcal{G}_T = \{\mathcal{G}_E + \mathcal{G}_C\}$, then if there exists both:
  − a minimal conditional independence $[X \perp\!\!\!\perp Y \mid \mathbf{Z}]$,
  − established absence of a causal path $Z \not\Rightarrow X$,
then there is a causal link (directed path) $Z \Rightarrow Y$ in $\mathcal{G}_C$.

The crucial observation is that these two pieces of information can be obtained from *different models*. In fact, the origin of the information $Z \not\Rightarrow X$ is irrelevant: be it from (in)dependencies via rule (2), other properties of the distribution, e.g. non-Gaussianity [11] or nonlinear features [12], or existing background knowledge. The only prerequisite for bringing results from various sources together is that the causal system at the centre is *invariant*, i.e. that the causal structure $\mathcal{G}_C$ remains the same across the different experiments $\mathcal{G}_T$, $\mathcal{G}'_T$ etc.

This result also shows why the well-known Y-structure: 4 nodes with $X \to Z \leftarrow W$ and $Z \to Y$, see [4], always enables identification of the causal link $Z \Rightarrow Y$: it is simply lemma 1 applied to overlapping nodes in a single model, in the form of rule (1) for $[X \perp\!\!\!\perp Y \mid Z]$, together with dependency $X \not\perp\!\!\!\perp W \mid Z$ created by $Z$ to eliminate $Z \not\Rightarrow X$ by rule (2).

## 5 Multiple models

In this article we focus on combining multiple conditional independence models represented by CPAGs. We want to use these models to convey as much about the underlying causal structure $\mathcal{G}_C$ as possible. We choose a *causal CPAG* as the target output model: similar in form and interpretation to a CPAG, where tails and arrowheads now represent *all known* (non)causal relations. This is not necessarily an equivalence class in accordance with the rules in [8], as it may contain more explicit information. Ingredients for extracting this information are the rules in theorem 1, in combination with the standard properties of causal relations: acyclic (if $X \Rightarrow Y$ then $Y \not\Rightarrow X$) and transitivity (if $X \Rightarrow Y$ and $Y \Rightarrow Z$ then $X \Rightarrow Z$). As the causal system is assumed invariant, the established (absence of) causal relations in one model are valid in all models.

A straightforward brute-force implementation is given by Algorithm 1. The input is a set of CPAG models $\mathcal{P}_i$, representing the conditional (in)dependence information between a set of observed variables, e.g. as learned by the extended FCI-algorithm [1, 8], from a number of different experiments $\mathcal{G}_T^{(i)}$ on an invariant causal system $\mathcal{G}_C$. The output is the single causal CPAG $\mathcal{G}$ over the *union* of all nodes in the input models $\mathcal{P}_i$.

---

**Input**   : set of CPAGs $\mathcal{P}_i$, fully $\circ\!-\!\circ$ connected graph $\mathcal{G}$
**Output** : causal graph $\mathcal{G}$
1: **for all** $\mathcal{P}_i$ **do**
2:     $\mathcal{G} \leftarrow$ eliminate all edges not appearing between nodes in $\mathcal{P}_i$                    ▷ Rule (3)
3:     $\mathcal{G} \leftarrow$ all definite (non)causal connections between nodes in $\mathcal{P}_i$          ▷ invariant structure
4: **end for**
5: **repeat**
6:     **for all** $\mathcal{P}_i$ **do**
7:         **for all** $\{X, Y, Z, \mathbf{W}\} \in \mathcal{P}_i$ **do**
8:             $\mathcal{G} \leftarrow (Z \not\Rightarrow \{X, Y, \mathbf{W}\})$, if $X \perp\!\!\!\perp Y \mid \mathbf{W}$ and $X \not\perp\!\!\!\perp Y \mid \{\mathbf{W} \cup Z\}$        ▷ Rule (2)
9:             $\mathcal{G} \leftarrow (Z \Rightarrow Y)$, if $[X \perp\!\!\!\perp Y \mid \{Z \cup \mathbf{W}\}]$ and $(Z \not\Rightarrow X) \in \mathcal{G}_C$          ▷ Rule (1)
10:        **end for**
11:    **end for**
12: **until** no more new non/causal information found

**Algorithm 1:** Brute force implementation of rules (1)-(3)

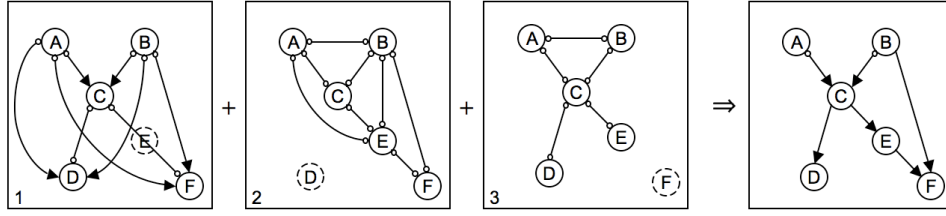

Figure 2: Three different experiments, one causal model

As an example, consider the three CPAG models on the l.h.s. of figure 2. None of these identifies a causal relation, yet despite the different (in)dependence relations, it is easily verified that the algorithm terminates after two loops with the nearly complete causal CPAG on the r.h.s. as the final output. Figure 1 shows corresponding experiments that explain the observed dependencies above.

To the best of our knowledge, Algorithm 1 is the first algorithm ever to perform such a derivation. Nevertheless, this brute-force approach exhibits a number of serious shortcomings. In the first place, the computational complexity of the repeated loop over all subsets in line 7 makes it not scalable: for small models like the ones in figure 2 the derivation is almost immediate, but for larger models it quickly becomes unfeasible. Secondly, for sparsely overlapping models, i.e. when the observed variables differ substantially between the models, the algorithm can miss certain relations: when a causal relation is found to be absent between two non-adjacent nodes, then this information cannot be recorded in $\mathcal{G}$, and subsequent causal information identifiable by rule (1) may be lost. These problems are addressed in the next section, resulting in the MCI-algorithm.

## 6 The MCI-algorithm

To tackle the computational complexity we first introduce the following notion: a path $\langle X, \ldots, Y \rangle$ in a CPAG is called a *possibly directed path* (or *p.d. path*) from $X$ to $Y$, if it can be converted into a directed path by changing circle marks into appropriate tails and arrowheads [6]. We can now state:

**Theorem 2.** Let $X$ and $Y$ be two variables in an experiment with causal structure $\mathcal{G}_T = \{\mathcal{G}_E + \mathcal{G}_C\}$, and let $\mathcal{P}_{[G]}$ be the corresponding CPAG over a subset of observed nodes from $\mathcal{G}_C$. Then the absence of a causal link $X \Rrightarrow Y$ is detectable from the conditional (in)dependence structure in this experiment iff there exists no p.d. path from $X$ to $Y$ in $\mathcal{P}_{[G]}$.

In other words: $X$ cannot be a cause (ancestor) of $Y$ if all paths from $X$ to $Y$ in the graph $\mathcal{P}_{[G]}$ go against an invariant arrowhead (signifying non-ancestorship) and vice versa. We refer to this as rule (4). Calculating which variables are connected by a p.d. path from a given CPAG is straightforward: turn the graph into a $\{0, 1\}$ adjacency matrix by setting all arrowheads to zero and all tails and circle marks to one, and compute the resulting reachability matrix. As this will uncover *all* detectable 'non-causal' relations in a CPAG in one go, it needs to be done only once for each model, and can be aggregated into a matrix $\mathbf{M}_C$ to make all tests for rule (2) in line 8 superfluous. If we also record all other established (non)causal relations in the matrix $\mathbf{M}_C$ as the algorithm progresses, then indirect causal relations are no longer lost when they cannot be transferred to the output graph $\mathcal{G}$. The next lemma propagates indirect (non)causal information from $\mathbf{M}_C$ to edge marks in the graph:

**Lemma 2.** Let $X$, $Y$ and $\mathbf{Z}$ be disjoint sets of variables in an experiment with causal structure $\mathcal{G}_T = \{\mathcal{G}_E + \mathcal{G}_C\}$, then for every $[X \perp\!\!\!\perp Y \mid \mathbf{Z}]$:
 – every (indirect) causal relation $X \Rightarrow Y$ implies causal links $\mathbf{Z} \Rightarrow Y$,
 – every (indirect) absence of causal relation $X \not\Rrightarrow Y$ implies no causal links $X \not\Rrightarrow \mathbf{Z}$.

The first makes it possible to orient indirect causal chains, the second shortens indirect non-causal links. We refer to these as rules (5) and (6), respectively. As a final improvement it is worth noting that for rules (1), (5) and (6) it is only relevant to know that a node $Z$ occurs in *some* $\mathbf{Z}$ in a minimal conditional independence relation $[X \perp\!\!\!\perp Y \mid \mathbf{Z}]$ separating $X$ and $Y$, but not what the other nodes in $\mathbf{Z}$ are or in what model(s) it occurred. We can introduce a structure $\mathbf{S}_{CI}$ to record all nodes $Z$ that occur in some minimal conditional independency in one of the models $\mathcal{P}_i$ for each combination of nodes $(X, Y)$, *before* any of the rules (1), (5) or (6) is processed. As a result, in the repeated causal inference loop no conditional independence / *m*-separation tests need to be performed at all.

```
      Input    : set of CPAGs 𝒫ᵢ, fully ∘−∘ connected graph 𝒢
      Output : causal graph 𝒢, causal relations matrix 𝐌_C
 1:  𝐌_C ← 𝟎                                                           ▷ no causal relations
 2:  for all 𝒫ᵢ do
 3:      𝒢 ← eliminate all edges not appearing between nodes in 𝒫ᵢ        ▷ Rule (3)
 4:      𝐌_C ← (X ⇏ Y), if no p.d. path ⟨X, . . . , Y⟩ ∈ 𝒫ᵢ               ▷ Rule (4)
 5:      𝐌_C ← (X ⇒ Y), if causal path ⟨X ⇒ . . . ⇒ Y⟩ ∈ 𝒫ᵢ              ▷ transitivity
 6:      for all (X, Y, Z) ∈ 𝒫ᵢ do
 7:          𝐒_{CI} ← triple (X, Y, Z), if Z ∈ 𝐙 for which [X ⊥⊥ Y | 𝐙]    ▷ combined SCI-matrix
 8:      end for
 9:  end for
10:  repeat
11:      for all (X, Y, Z) ∈ 𝒢 do
12:          𝐌_C ← (Z ⇒ Y), for unused (X, Y, Z) ∈ 𝐒_{CI} with (Z ⇏ X) ∈ 𝐌_C    ▷ Rule (1)
13:          𝐌_C ← (X ⇏ Z), for unused (X ⇏ Y) ∈ 𝐌_C with (X, Y, Z) ∈ 𝐒_{CI}    ▷ Rule (5)
14:          𝐌_C ← (Z ⇒ Y), for unused (X ⇒ Y) ∈ 𝐌_C with (X, Y, Z) ∈ 𝐒_{CI}    ▷ Rule (6)
15:      end for
16:  until no more new causal information found
17:  𝒢 ← non/causal info in 𝐌_C                                          ▷ tails/arrowheads
```

**Algorithm 2:** MCI algorithm

With these results we can now give an improved version of the brute-force approach: the *Multiple model Causal Inference* (MCI) algorithm, above. The input is still a set of CPAG models from different experiments, but the output is now twofold: the graph $\mathcal{G}$, containing the causal structure uncovered for the underlying system $\mathcal{G}_C$, as well as the matrix $\mathbf{M}_C$ with an explicit representation of all (non)causal relations between observed variables, including remaining indirect information that cannot be read from the graph $\mathcal{G}$.

The first stage (lines 2-9) is a pre-processing step to extract all necessary information for the second stage from each of the models separately. Building the $\mathbf{S}_{CI}$ matrix is the most expensive step as it involves testing for conditional independencies (*m*-separation) for increasing sets of variables. This can be efficiently implemented by noting that nodes connected by an edge will not be separated and that many other combinations will not have to be tested as they contain a subset for which a (minimal) conditional independency has already been established. If a (non)causal relation is found between adjacent variables in $\mathcal{G}$, or one that can be used to infer other intermediate relations (lines 13-14), then it can be marked as 'processed' to avoid unnecessary checks. Similar for the entries recorded in the minimal conditional independence structure $\mathbf{S}_{CI}$.

The MCI algorithm is provably sound in the sense that if all input CPAG models $\mathcal{P}_i$ are valid, then all (absence of) causal relations identified by the algorithm in the output graph $\mathcal{G}$ and (non)causal relations matrix $\mathbf{M}_C$ are also valid, provided that the causal system $\mathcal{G}_C$ is an invariant causal DAG and the causal faithfulness assumption is satisfied.

## 7  Experimental results

We tested the MCI-algorithm on a variety of synthetic data sets to verify its validity and assess its behaviour and performance in uncovering causal information from multiple models. For the generation of random causal DAGs we used a variant of [13] to control the distribution of edges over nodes in the network. The random experiments in each run were generated from this causal DAG by including a random context and hidden nodes. For each network the corresponding CPAG was computed, and together used as the set of input models for the MCI-algorithm. The generated output $\mathcal{G}$ and $\mathbf{M}_C$ was verified against the true causal DAG and expressed as a percentage of the true number of (non-)causal relations.

To assess the performance we introduced two reference methods to act as a benchmark for the MCI-algorithm (in the absence of other algorithms that can validly handle different contexts). The first is a common sense method, indicated as 'sum-FCI', that utilizes the transitive closure of all causal relations in the input CPAGs, that could have been identified by FCI in the large sample limit. As the

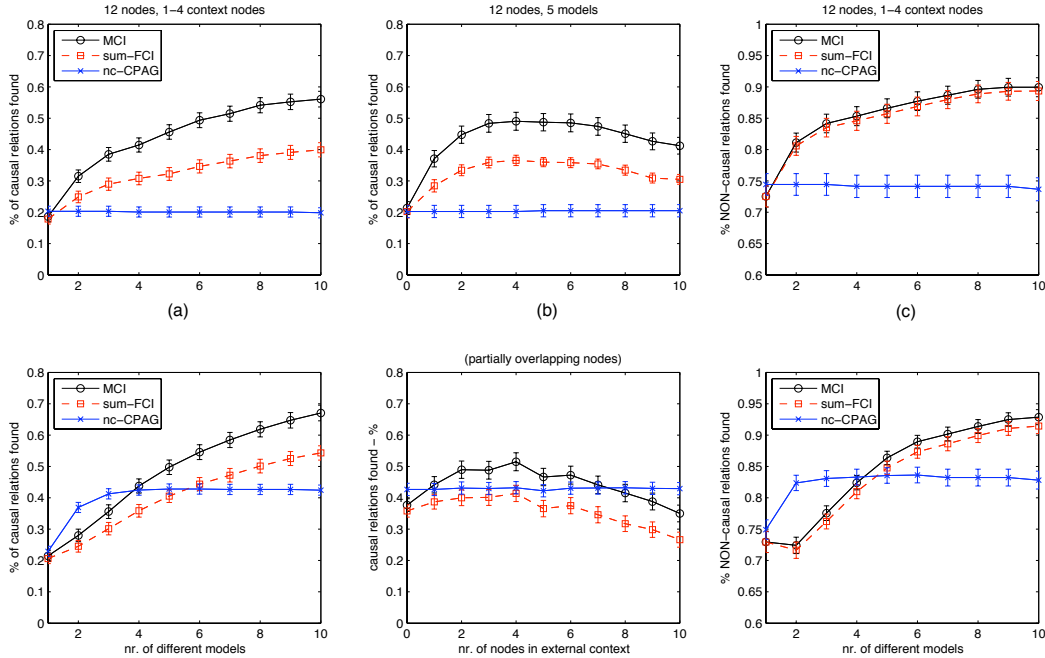

Figure 3: Proportion of causal relations discovered by the MCI-algorithm in different settings; (a) causal relations vs. nr. of models, (b) causal relations vs. nr. of context nodes, (c) non-causal relations ⇝ vs. nr. of models; (top) identical observed nodes in input models, (bottom) only partially overlapping observed nodes

second benchmark we take all causal information contained in the CPAG over the union of observed variables, independent of the context, hence 'nc-CPAG' for 'no context'. Note that this is not really a method as it uses information directly derived from the true causal graph.

In figure 3, each graph depicts the percentage of causal (a&b) or non-causal (c) relations uncovered by each of the three methods: MCI, sum-FCI and nc-CPAG, as a function of the number of input models (a&c) or the number of nodes in the context (b), averaged over 200 runs, for both identical (top) or only partly overlapping (bottom) observed nodes in the input models. Performance is calculated as the proportion of uncovered relations as compared to the actual number of non/causal relations in the true causal graph over the union of observed nodes in each model set. In these runs the underlying causal graphs contained 12 nodes with edge degree $\leq 5$. Tests for other, much sparser/denser graphs up to 20 nodes, showed comparable results.

Some typical behaviour is easily recognized: MCI always outperforms sum-FCI, and more input models always improve performance. Also non-causal information (c) is much easier to find than definite causal relations (a&b). For single models / no context the performance of all three methods is very similar, although not necessarily identical. The perceived initial drop in performance in fig.3(c,bottom) is only because in going to two models the number of non-causal relations in the union rises more quickly than the number of new relations that is actually found (due to lack of overlap). A striking result that is clearly brought out is that adding random context actually *improves* detection rate of causal relations. The rationale behind this effect is that externally induced links can introduce conditional dependencies, allowing the deduction of non-causal links that are not otherwise detectable; these in turn may lead to other causal relations that can be inferred, and so on. If the context is expanded further, at some point the detection rate will start to deteriorate as the causal structure will be swamped by the externally induced links (b).

We want to stress that for the many tens of thousands of (non)causal relations identified by the MCI algorithm in all the runs, not a single one was found to be invalid on comparing with the true causal graph. For $\gtrsim 8$ nodes the algorithm spends the majority of its time building the $\mathbf{S}_{CI}$ matrix in lines 6-8. The actual number of minimal conditional independencies found, however, is quite low, typically in the order of a few dozen for graphs of up to 12 nodes.

# 8 Conclusion

We have shown the first principled algorithm that can use results from *different* experiments to uncover new (non)causal information. It is provably sound in the large sample limit, provided the input models are learned by a valid algorithm like the FCI algorithm with CPAG extension [8]. In its current implementation the MCI-algorithm is a fast and practical method that can easily be applied to sets of models of up to 20 nodes. Compared to related algorithms like ION, it produces very concise and easily interpretable output, and does not suffer from the inability to handle any differences in observed dependencies between data sets [3]. For larger models it can be converted into an anytime algorithm by running over minimal conditional independencies from subsets of increasing size: at each level all uncovered causal information is valid, and, for reasonably sparse models, most will already be found at low levels. For very large models an exciting possibility is to target only specific causal relations: finding the right combination of (in)dependencies is sufficient to decide if it is causal, even when there is no hope of deriving a global CPAG model.

From the construction of the MCI-algorithm it is sound, but not necessarily complete. Preliminary results show that theorem 1 already covers all invariant arrowheads in the single model case [8], and suggest that an additional rule is sufficient to cover all tails as well. We aim to extend this result to the multiple model domain. Integrating our approach with recent developments in causal discovery that are not based on independence constraints [11, 12] can provide for even more detectable causal information. When applied to real data sets the large sample limit no longer applies and inconsistent causal relations may result. It should be possible to exclude the contribution of such links on the final output. Alternatively, output might be generalized to quantities like 'the probability of a causal relation' based on the strength of appropriate conditional (in)dependencies in the available data.

**Acknowledgement** This research was supported by VICI grant 639.023.604 from the Netherlands Organization for Scientific Research (NWO).

## Appendix - proof outlines

**Theorem 1 / Lemma 1** (*for details see also [14]*)

(1) Without selection bias, nodes $X$ and $Y$ are dependent iff they are connected by treks in $\mathcal{G}_T$. A node $Z \in \mathbf{Z}$ that blocks such a trek has a directed path in $\mathcal{G}_C$ to $X$ and/or $Y$, but can unblock other paths. These paths contain a trek between $Z$ and $X/Y$, and must blocked by a node $Z' \in \mathbf{Z}_{\setminus Z}$, which therefore also has a causal link to $X$ or $Y$ (possibly via $Z$). $Z'$ in turn can unblock other paths, etc. Minimality guarantees that it holds for all nodes in $\mathbf{Z}$. Eliminating $Z \not\Rightarrow X$ then leaves $Z \Rightarrow Y$ as the only option (lemma 1).

(2) To create the dependency, $W$ must be a (descendant of a) collider between two unblocked paths $\pi_1 = \langle X, \ldots, W \rangle$ and $\pi_2 = \langle Y, \ldots, W \rangle$ given $\mathbf{Z}$. Any directed path from $W$ to a $Z \in \mathbf{Z}$ implies that conditioning on $W$ is not needed when already conditioning on $\mathbf{Z}$. In combination with $\pi_1$ or $\pi_2$, a directed path from $W$ to $X$ or $Y$ in $\mathcal{G}_C$ would make $W$ a noncollider on an unblocked path between $X$ and $Y$ given $\mathbf{Z}$, contrary to $X \perp\!\!\!\perp Y \mid \mathbf{Z}$.

(3) A directed path between $X$ and $Y$ that is not blocked by $\mathbf{Z}$ would result in $X \not\perp\!\!\!\perp Y \mid \mathbf{Z}$, see [1].

**Theorem 2**
'$\Leftarrow$' follows from the fact that a directed path $\pi = \langle X, \ldots, Y \rangle$ in the underlying causal DAG $\mathcal{G}_C$ implies existence of a directed path in the true MAG over the observed nodes and therefore at least the existence of a p.d. path in the CPAG $\mathcal{P}_{[G]}$.
'$\Rightarrow$' follows from the completeness of the CPAG in combination with theorem 2 in [8] about orientability of CPAGs into MAGs. This, together with Meek's algorithm [15] for orienting chordal graphs into DAGs with no unshielded colliders, shows that it is always possible to turn a p.d. path into a directed path in a MAG that is a member of the equivalence class $\mathcal{P}_{[G]}$. Therefore, a p.d. path from $X$ to $Y$ in $\mathcal{P}_{[G]}$ implies there is at least some underlying causal DAG in which it is a causal path, and so cannot correspond to a valid, detectable absence of a causal link. $\square$

**Lemma 2**
From rule (1) in Theorem 1, $[X \perp\!\!\!\perp Y \mid \mathbf{Z}]$ implies causal links $\mathbf{Z} \Rightarrow X$ and/or $\mathbf{Z} \Rightarrow Y$. If $X \Rightarrow Y$ then by transitivity $\mathbf{Z} \Rightarrow X$ also implies $\mathbf{Z} \Rightarrow Y$. If $X \not\Rightarrow Y$ then for any $Z \in \mathbf{Z}$, $X \Rightarrow Z$ implies $Z \Rightarrow Y$ and so (transitivity) also $X \Rightarrow Y$, in contradiction of the given; therefore $X \not\Rightarrow \mathbf{Z}$. $\square$

# References

[1] P. Spirtes, C. Glymour, and R. Scheines, *Causation, Prediction, and Search*. Cambridge, Massachusetts: The MIT Press, 2nd ed., 2000.

[2] D. Chickering, "Optimal structure identification with greedy search," *Journal of Machine Learning Research*, vol. 3, no. 3, pp. 507–554, 2002.

[3] R. Tillman, D. Danks, and C. Glymour, "Integrating locally learned causal structures with overlapping variables," in *Advances in Neural Information Processing Systems, 21*, 2008.

[4] S. Mani, G. Cooper, and P. Spirtes, "A theoretical study of Y structures for causal discovery," in *Proceedings of the 22nd Conference in Uncertainty in Artificial Intelligence*, pp. 314–323, 2006.

[5] J. Pearl, *Causality: models, reasoning and inference*. Cambridge University Press, 2000.

[6] J. Zhang, "Causal reasoning with ancestral graphs," *Journal of Machine Learning Research*, vol. 9, pp. 1437 – 1474, 2008.

[7] T. Richardson and P. Spirtes, "Ancestral graph Markov models," *Ann. Stat.*, vol. 30, no. 4, pp. 962–1030, 2002.

[8] J. Zhang, "On the completeness of orientation rules for causal discovery in the presence of latent confounders and selection bias," *Artificial Intelligence*, vol. 172, no. 16-17, pp. 1873 – 1896, 2008.

[9] J. Zhang and P. Spirtes, "Detection of unfaithfulness and robust causal inference," *Minds and Machines*, vol. 2, no. 18, pp. 239–271, 2008.

[10] P. Spirtes, C. Meek, and T. Richardson, "An algorithm for causal inference in the presence of latent variables and selection bias," in *Computation, Causation, and Discovery*, pp. 211–252, 1999.

[11] S. Shimizu, P. Hoyer, A. Hyvärinen, and A. Kerminen, "A linear non-Gaussian acyclic model for causal discovery," *Journal of Machine Learning Research*, vol. 7, pp. 2003–2030, 2006.

[12] P. Hoyer, D. Janzing, J. Mooij, J. Peters, and B. Schölkopf, "Nonlinear causal discovery with additive noise models," in *Advances in Neural Information Processing Systems 21 (NIPS*2008)*, pp. 689–696, 2009.

[13] J. Ide and F. Cozman, "Random generation of Bayesian networks," in *Advances in Artificial Intelligence*, pp. 366–376, Springer Berlin, 2002.

[14] T. Claassen and T. Heskes, "Learning causal network structure from multiple (in)dependence models," in *Proceedings of the Fifth European Workshop on Probabilistic Graphical Models*, 2010.

[15] C. Meek, "Causal inference and causal explanation with background knowledge," in *UAI*, pp. 403–410, Morgan Kaufmann, 1995.

